# Ranking annotators for crowdsourced labeling tasks

**Vikas C. Raykar**
Siemens Healthcare, Malvern, PA, USA
vikas.raykar@siemens.com

**Shipeng Yu**
Siemens Healthcare, Malvern, PA, USA
shipeng.yu@siemens.com

## Abstract

With the advent of crowdsourcing services it has become quite cheap and reasonably effective to get a dataset labeled by multiple annotators in a short amount of time. Various methods have been proposed to estimate the consensus labels by correcting for the bias of annotators with different kinds of expertise. Often we have low quality annotators or *spammers*–annotators who assign labels randomly (*e.g.*, without actually looking at the instance). Spammers can make the cost of acquiring labels very expensive and can potentially degrade the quality of the consensus labels. In this paper we formalize the notion of a spammer and define a score which can be used to rank the annotators—with the spammers having a score close to zero and the good annotators having a high score close to one.

## 1 Spammers in crowdsourced labeling tasks

Annotating an unlabeled dataset is one of the bottlenecks in using supervised learning to build good predictive models. Getting a dataset labeled by experts can be expensive and time consuming. With the advent of crowdsourcing services (Amazon's Mechanical Turk being a prime example) it has become quite easy and inexpensive to acquire labels from a large number of annotators in a short amount of time (see [8], [10], and [11] for some computer vision and natural language processing case studies). One drawback of most crowdsourcing services is that we do not have tight control over the quality of the annotators. The annotators can come from a diverse pool including genuine experts, novices, biased annotators, malicious annotators, and spammers. Hence in order to get good quality labels requestors typically get each instance labeled by multiple annotators and these multiple annotations are then consolidated either using a simple majority voting or more sophisticated methods that model and correct for the annotator biases [3, 9, 6, 7, 14] and/or task complexity [2, 13, 12].

In this paper we are interested in ranking annotators based on how *spammer* like each annotator is. In our context a spammer is a low quality annotator who assigns random labels (maybe because the annotator does not understand the labeling criteria, does not look at the instances when labeling, or maybe a bot pretending to be a human annotator). Spammers can significantly *increase the cost* of acquiring annotations (since they need to be paid) and at the same time *decrease the accuracy* of the final consensus labels. A mechanism to detect and eliminate spammers is a desirable feature for any crowdsourcing market place. For example one can give monetary bonuses to good annotators and deny payments to spammers.

The main contribution of this paper is to formalize the notion of a spammer for binary, categorical, and ordinal labeling tasks. More specifically we define a *scalar metric* which can be used to *rank the annotators*—with the spammers having a score close to zero and the good annotators having a score close to one (see Figure 4). We summarize the multiple parameters corresponding to each annotator into a single score indicative of how spammer like the annotator is. While this spammer score was implicit for binary labels in earlier works [3, 9, 2, 6] the extension to categorical and ordinal labels is novel and is quite different from the accuracy computed from the confusion rate matrix. An attempt to quantify the quality of the workers based on the confusion matrix was recently made by [4] where they transformed the observed labels into posterior soft labels based on the estimated confusion

matrix. While we obtain somewhat similar annotator rankings, we differ from this work in that our score is directly defined in terms of the annotator parameters (see § 5 for more details).

The rest of the paper is organized as follows. For ease of exposition we start with binary labels (§ 2) and later extend it to categorical (§ 3) and ordinal labels (§ 4). We first specify the annotator model used, formalize the notion of a spammer, and propose an appropriate score in terms of the annotator model parameters. We do not dwell too much on the estimation of the annotator model parameters. These parameters can either be estimated directly using known gold standard [1] or the iterative algorithms that estimate the annotator model parameters without actually knowing the gold standard [3, 9, 2, 6, 7]. In the experimental section (§ 6) we obtain rankings for the annotators using the proposed spammer scores on some publicly available data from different domains.

## 2  Spammer score for crowdsourced binary labels

**Annotator model** Let $y_i^j \in \{0, 1\}$ be the label assigned to the $i^{\text{th}}$ instance by the $j^{\text{th}}$ annotator, and let $y_i \in \{0, 1\}$ be the actual (unobserved) binary label. We model the accuracy of the annotator separately on the positive and the negative examples. If the true label is one, the *sensitivity* (true positive rate) $\alpha^j$ for the $j^{\text{th}}$ annotator is defined as the probability that the annotator labels it as one.

$$\alpha^j := \Pr[y_i^j = 1 | y_i = 1].$$

On the other hand, if the true label is zero, the *specificity* ($1-$false positive rate) $\beta^j$ is defined as the probability that annotator labels it as zero.

$$\beta^j := \Pr[y_i^j = 0 | y_i = 0].$$

Extensions of this basic model have been proposed to include item level difficulty [2, 13] and also to model the annotator performance based on the feature vector [14]. For simplicity we use the basic model proposed in [7] in our formulation. Based on many instances labeled by multiple annotators the maximum likelihood estimator for the annotator parameters $(\alpha^j, \beta^j)$ and also the consensus ground truth $(y_i)$ can be estimated iteratively [3, 7] via the Expectation Maximization (EM) algorithm. The EM algorithm iteratively establishes a particular gold standard (initialized via majority voting), measures the performance of the annotators given that gold standard (M-step), and refines the gold standard based on the performance measures (E-step).

**Who is a spammer?** Intuitively, *a spammer assigns labels randomly*—maybe because the annotator does not understand the labeling criteria, does not look at the instances when labeling, or maybe a bot pretending to be a human annotator. More precisely an annotator is a spammer if the probability of observed label $y_i^j$ being one given the true label $y_i$ is independent of the true label, *i.e.*,

$$\Pr[y_i^j = 1 | y_i] = \Pr[y_i^j = 1]. \tag{1}$$

This means that the annotator is assigning labels randomly by flipping a coin with bias $\Pr[y_i^j = 1]$ without actually looking at the data. Equivalently (1) can be written as

$$\Pr[y_i^j = 1 | y_i = 1] = \Pr[y_i^j = 1 | y_i = 0] \quad \text{which implies} \quad \alpha^j = 1 - \beta^j. \tag{2}$$

Hence in the context of the annotator model defined earlier a perfect spammer is an annotator for whom $\alpha^j + \beta^j - 1 = 0$. This corresponds to the diagonal line on the Receiver Operating Characteristic (ROC) plot (see Figure 1(a)) [2]. If $\alpha^j + \beta^j - 1 < 0$ then the annotators lies below the diagonal line and is a malicious annotator who flips the labels. Note that a malicious annotator has discriminatory power if we can detect them and flip their labels. In fact the methods proposed in [3, 7] can automatically flip the labels for the malicious annotators. Hence we define the spammer score for an annotator as

$$\boxed{\mathcal{S}^j = (\alpha^j + \beta^j - 1)^2} \tag{3}$$

An annotator is a spammer if $\mathcal{S}^j$ is close to zero. Good annotators have $\mathcal{S}^j > 0$ while a perfect annotator has $\mathcal{S}^j = 1$.

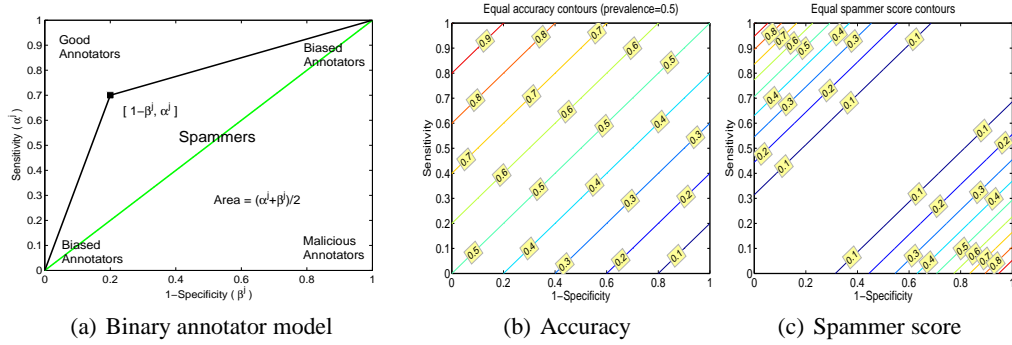

(a) Binary annotator model  (b) Accuracy  (c) Spammer score

Figure 1: (a) For binary labels an annotator is modeled by his/her sensitivity and specificity. A perfect spammer lies on the diagonal line on the ROC plot. (b) Contours of equal accuracy (4) and (c) equal spammer score (3).

**Accuracy** This notion of a spammer is quite different for that of the *accuracy* of an annotator. An annotator with high accuracy is a good annotator but one with low accuracy is not necessarily a spammer. The accuracy is computed as

$$\text{Accuracy}^j = \Pr[y_i^j = y_i] = \sum_{k=0}^{1} \Pr[y_i^j = 1 | y_i = k]\Pr[y_i = k] = \alpha^j p + \beta^j(1 - p), \qquad (4)$$

where $p := \Pr[y_i = 1]$ is the prevalence of the positive class. Note that accuracy depends on prevalence. Our proposed spammer score does not depend on prevalence and essentially quantifies the annotator's inherent discriminatory power. Figure 1(b) shows the contours of equal accuracy on the ROC plot. Note that annotators below the diagonal line (malicious annotators) have low accuracy. The malicious annotators are good annotators but they flip their labels and as such are not spammers if we can detect them and then correct for the flipping. In fact the EM algorithms [3, 7] can correctly flip the labels for the malicious annotators and hence they should not be treated as spammers. Figure 1(c) also shows the contours of equal score for our proposed score and it can be seen that the malicious annotators have a high score and only annotators along the diagonal have a low score (spammers).

**Log-odds** Another interpretation of a spammer can be seen from the log odds. Using Bayes' rule the posterior log-odds can be written as

$$\log \frac{\Pr[y_i = 1 | y_i^j]}{\Pr[y_i = 0 | y_i^j]} = \log \frac{\Pr[y_i^j | y_i = 1]}{\Pr[y_i^j | y_i = 0]} + \log \frac{p}{1 - p}.$$

If an annotator is a spammer (*i.e.*, (2) holds) then $\log \frac{\Pr[y_i = 1 | y_i^j]}{\Pr[y_i = 0 | y_i^j]} = \log \frac{p}{1-p}$. Essentially the annotator provides no information in updating the posterior log-odds and hence does not contribute to the estimation of the actual true label.

## 3 Spammer score for categorical labels

**Annotator model** Suppose there are $K \geq 2$ categories. We introduce a multinomial parameter $\boldsymbol{\alpha}_c^j = (\alpha_{c1}^j, \ldots, \alpha_{cK}^j)$ for each annotator, where

$$\alpha_{ck}^j := \Pr[y_i^j = k | y_i = c] \qquad \text{and} \qquad \sum_{k=1}^{K} \alpha_{ck}^j = 1.$$

The term $\alpha_{ck}^j$ denotes the probability that annotator $j$ assigns class $k$ to an instance given that the true class is $c$. When $K = 2$, $\alpha_{11}^j$ and $\alpha_{00}^j$ are sensitivity and specificity, respectively.

**Who is a spammer?** As earlier a spammer assigns labels randomly, *i.e.*,

$$\Pr[y_i^j = k | y_i] = \Pr[y_i^j = k], \forall k.$$

This is equivalent to $\Pr[y_i^j = k | y_i = c] = \Pr[y_i^j = k | y_i = c'], \forall c, c', k = 1, \ldots, K$— which means knowing the true class label being $c$ or $c'$ does not change the probability of the annotator's assigned label. This indicates that the annotator $j$ is a spammer if

$$\alpha_{ck}^j = \alpha_{c'k}^j, \forall c, c', k = 1, \ldots, K. \tag{5}$$

Let $\mathbf{A}^j$ be the $K \times K$ confusion rate matrix with entries $[\mathbf{A}^j]_{ck} = \alpha_{ck}$—a spammer would have all the rows of $\mathbf{A}^j$ equal, for example, $\mathbf{A}^j = \begin{bmatrix} 0.50 & 0.25 & 0.25 \\ 0.50 & 0.25 & 0.25 \\ 0.50 & 0.25 & 0.25 \end{bmatrix}$, for a three class categorical annotation problem. Essentially $\mathbf{A}^j$ is a rank one matrix of the form $\mathbf{A}^j = \mathbf{e}\mathbf{v}_j^\top$, for some column vector $\mathbf{v}_j \in \mathbb{R}^K$ that satisfies $\mathbf{v}_j^\top \mathbf{e} = 1$, where $\mathbf{e}$ is column vector of ones.

In the binary case we had this natural notion of spammer as an annotator for whom $\alpha^j + \beta^j - 1$ was close to zero. One natural way to summarize (5) would be in terms of the distance (Frobenius norm) of the confusion matrix to the closest rank one approximation, $i.e$,

$$\mathcal{S}^j := \|\mathbf{A}^j - \mathbf{e}\hat{\mathbf{v}}_j^\top\|_F^2, \tag{6}$$

where $\hat{\mathbf{v}}_j$ solves

$$\hat{\mathbf{v}}_j = \arg\min_{\mathbf{v}_j} \|\mathbf{A}^j - \mathbf{e}\mathbf{v}_j^\top\|_F^2 \qquad s.t. \quad \mathbf{v}_j^\top \mathbf{e} = 1. \tag{7}$$

Solving (7) yields $\hat{\mathbf{v}}_j = (1/K)\mathbf{A}^{j\top}\mathbf{e}$, which is the mean of the rows of $\mathbf{A}^j$. Then from (6) we have

$$\mathcal{S}^j = \left\| \left( \mathbf{I} - \frac{1}{K}\mathbf{e}\mathbf{e}^\top \right) \mathbf{A}^j \right\|_F^2 = \frac{1}{K} \sum_{c<c'} \sum_k (\alpha_{ck}^j - \alpha_{c'k}^j)^2.$$

So a spammer is an annotator for whom $\mathcal{S}^j$ is close to zero. A perfect annotator has $\mathcal{S}^j = K - 1$. We normalize this score to lie between 0 and 1.

$$\boxed{\mathcal{S}^j = \frac{1}{K(K-1)} \sum_{c<c'} \sum_k (\alpha_{ck}^j - \alpha_{c'k}^j)^2} \tag{8}$$

When $K = 2$ this is equivalent to the score proposed earlier for binary labels. As earlier this notion of a spammer is different than the accuracy computed from the confusion rate matrix and the prevalence. The accuracy is computed as $\text{Accuracy}^j = \Pr[y_i^j = y_i] = \sum_{k=1}^K \Pr[y_i^j = k | y_i = k]\Pr[y_i = k] = \sum_{k=1}^K \alpha_{kk}^j \Pr[y_i = k]$.

## 4  Spammer score for ordinal labels

A commonly used paradigm to annotate instances is to use *ordinal scales* where an annotator is asked to rate an instance on a certain ordinal scale, say $\{1, \ldots, K\}$. For example, rating a restaurant on a scale of 1 to 5 or assessing the malignancy of a lesion on a BIRADS scale of 1 to 5 for mammography. This differs from categorical labels where there is no order among the multiple class labels. An ordinal variable expresses rank and there is an implicit ordering $1 < \ldots < K$.

**Annotator model** It is conceptually easier to think of the true label to be binary, that is, $y_i \in \{0, 1\}$. For example in mammography a lesion is either malignant (1) or benign (0) (which can be confirmed by biopsy) and the BIRADS ordinal scale is a means for the radiologist to quantify the uncertainty based on the digital mammogram. The radiologist assigns a higher value of the label if he/she thinks the true label is closer to one. As earlier we characterize each annotator by the sensitivity and the specificity, but the main difference is that we now define the sensitivity and specificity for each ordinal label (or threshold) $k \in \{1, \ldots, K\}$. Let $\alpha_k^j$ and $\beta_k^j$ be the sensitivity and specificity respectively of the $j^{th}$ annotator corresponding to the threshold $k$, that is,

$$\alpha_k^j = \Pr[y_i^j \geq k \mid y_i = 1] \quad \text{and} \quad \beta_k^j = \Pr[y_i^j < k \mid y_i = 0].$$

Note that $\alpha_1^j = 1$, $\beta_1^j = 0$ and $\alpha_{K+1}^j = 0$, $\beta_{K+1}^j = 1$ from this definition. Hence each annotator is parameterized by a set of $2(K-1)$ parameters $[\alpha_2^j, \beta_2^j, \ldots, \alpha_K^j, \beta_K^j]$. This corresponds to an empirical ROC curve for the annotator (Figure 2).

**Who is a spammer?** As earlier we define an annotator $j$ to be a spammer if $\Pr[y_i^j = k | y_i = 1] = \Pr[y_i^j = k | y_i = 0] \; \forall k = 1, \ldots, K$. Note that from the annotation model we have [3] $\Pr[y_i^j = k \mid y_i = 1] = \alpha_k^j - \alpha_{k+1}^j$ and $\Pr[y_i^j = k \mid y_i = 0] = \beta_{k+1}^j - \beta_k^j$. This implies that annotator $j$ is a spammer if $\alpha_k^j - \alpha_{k+1}^j = \beta_{k+1}^j - \beta_k^j, \forall k = 1, \ldots, K$, which leads to $\alpha_k^j + \beta_k^j = \alpha_1^j + \beta_1^j = 1, \forall k$. This means that for every $k$, the point $(1 - \beta_k^j, \alpha_k^j)$ lies on the diagonal line in the ROC plot shown in Figure 2. The area under the empirical ROC curve can be computed as (see Figure 2) $\mathrm{AUC}^j = \frac{1}{2} \sum_{k=1}^{K} (\alpha_{k+1}^j + \alpha_k^j)(\beta_{k+1}^j - \beta_k^j)$, and can be used to define the following spammer score as $(2\mathrm{AUC}^j - 1)^2$ to rank the different annotators.

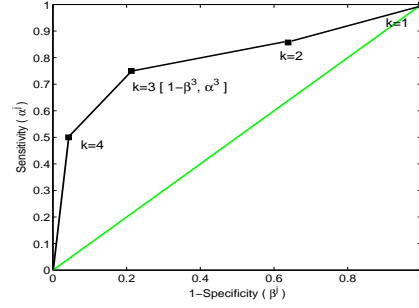

Figure 2: Ordinal labels: An annotator is modeled by sensitivity/specificity for each threshold.

$$\mathcal{S}^j = \left( \left[ \sum_{k=1}^{K} (\alpha_{k+1}^j + \alpha_k^j)(\beta_{k+1}^j - \beta_k^j) \right] - 1 \right)^2 \tag{9}$$

With two levels this expression defaults to the binary case. An annotator is a spammer if $\mathcal{S}^j$ is close to zero. Good annotators have $\mathcal{S}^j > 0$ while a perfect annotator has $\mathcal{S}^j = 1$.

## 5 Previous work

Recently Ipeirotis *et.al.* [4] proposed a score for categorical labels based on the expected cost of the posterior label. In this section we briefly describe their approach and compare it with our proposed score. For each instance labeled by the annotator they first compute the posterior (soft) label $\Pr[y_i = c | y_i^j]$ for $c = 1, \ldots, K$, where $y_i^j$ is the label assigned to the $i^{th}$ instance by the $j^{th}$ annotator and $y_i$ is the true unknown label. The posterior label is computed via Bayes' rule as $\Pr[y_i = c | y_i^j] \propto \Pr[y_i^j | y_i = c] \Pr[y_i = c] = (\alpha_{ck}^j)^{\delta(y_i^j, k)} p_c$, where $p_c = \Pr[y_i = c]$ is the prevalence of class $c$. The score for a spammer is based on the intuition that the posterior label vector $(\Pr[y_i = 1 | y_i^j], \ldots, \Pr[y_i = K | y_i^j])$ for a good annotator will have all the probability mass concentrated on single class. For example for a three class problem (with equal prevalence), a posterior label vector of $(1, 0, 0)$ (certain that the class is one) comes from a good annotator while a $(1/3, 1/3, 1/3)$ (complete uncertainty about the class label) comes from spammer. Based on this they define the following score for each annotator

$$\mathrm{Score}^j = \frac{1}{N} \sum_{i=1}^{N} \left[ \sum_{c=1}^{K} \sum_{k=1}^{K} \left( \mathrm{cost}_{ck} \Pr[y_i = k | y_i^j] \Pr[y_i = c | y_i^j] \right) \right]. \tag{10}$$

where $\mathrm{cost}_{ck}$ is the misclassification cost when an instance of class $c$ is classified as $k$. Essentially this is capturing some sort of uncertainty of the posterior label averaged over all the instances. Perfect workers have a score $\mathrm{Score}^j = 0$ while spammers will have high score. An entropic version of this score based on similar ideas has also been recently proposed in [5]. Our proposed spammer score differs from this approach in the following aspects: (1) Implicit in the score defined above (10) is the assumption that an annotator is a spammer when $\Pr[y_i = c | y_i^j] = \Pr[y_i = c]$, *i.e.*, the estimated posterior labels are simply based on the prevalence and do not depend on the observed labels. By Bayes' rule this is equivalent to $\Pr[y_i^j | y_i = c] = \Pr[y_i^j]$ which is what we have used to define our spammer score. (2) While both notions of a spammer are equivalent, the approach of [4] first computes the posterior labels based on the observed data, the class prevalence and the annotator

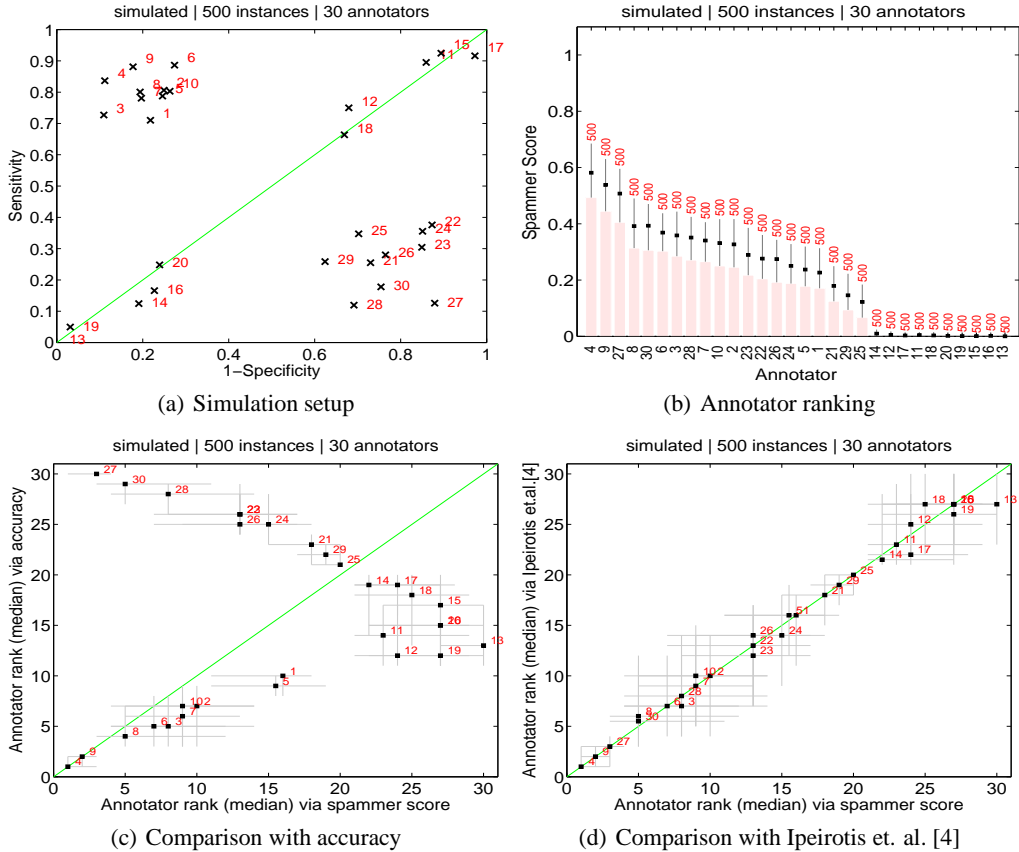

Figure 3: (a) The simulation setup consisting of 10 good annotators (annotators 1 to 10), 10 spammers (11 to 20), and 10 malicious annotators (21 to 30). (b) The ranking of annotators obtained using the proposed spammer score. The spammer score ranges from 0 to 1, the lower the score, the more spammy the annotator. The mean spammer score and the 95% confidence intervals (CI) are shown—obtained from 100 bootstrap replications. The annotators are ranked based on the lower limit of the 95% CI. The number at the top of the CI bar shows the number of instances annotated by that annotator. (c) and (d) Comparison of the median rank obtained via the spammer score with the rank obtained using (c) accuracy and (d) the method proposed by Ipeirotis et. al. [4].

parameters and then computes the expected cost. Our proposed spammer score does not depend on the prevalence of the class. Our score is also directly defined only in terms of the annotator confusion matrix and does not need the observed labels. (3) For the score defined in (10) while perfect annotators have a score of 0 it is not clear what should be a good baseline for a spammer. The authors suggest to compute the baseline by assuming that a worker assigns as label the class with maximum prevalence. Our proposed score has a natural scale with a perfect annotator having a score of 1 and a spammer having a score of 0. (4) However one advantage of the approach in [4] is that they can directly incorporate varied misclassification costs.

## 6    Experiments

**Ranking annotators based on the confidence interval** As mentioned earlier the annotator model parameters can be estimated using the iterative EM algorithms [3, 7] and these estimated annotator parameters can then be used to compute the spammer score. The spammer score can then be used to rank the annotators. However one commonly observed phenomenon when working with crowd-sourced data is that we have a lot of annotators who label only a very few instances. As a result the annotator parameters cannot be reliably estimated for these annotators. In order to factor this uncertainty in the estimation of the model parameters we compute the spammer score for 100 bootstrap replications. Based on this we compute the 95% confidence intervals (CI) for the spammer score for each annotator. We rank the annotators based on the lower limit of the 95% CI. The CIs are wider

Table 1: *Datasets* $N$ is the number of instances. $M$ is the number of annotators. $M^*$ is the mean/median number of annotators per instance. $N^*$ is the mean/median number of instances labeled by each annotator.

| Dataset | Type | $N$ | $M$ | $M^*$ | $N^*$ | Brief Description |
|---|---|---|---|---|---|---|
| bluebird | binary | 108 | 39 | 39/39 | 108/108 | *bird identification* [12] The annotator had to identify whether there was an *Indigo Bunting* or *Blue Grosbeak* in the image. |
| temp | binary | 462 | 76 | 10/10 | 61/16 | *event annotation* [10] Given a dialogue and a pair of verbs annotators need to label whether the event described by the first verb occurs before or after the second. |
| wsd | categorical/3 | 177 | 34 | 10/10 | 52/20 | *word sense disambiguation* [10] The labeler is given a paragraph of text containing the word "president" and asked to label one of the three appropriate senses. |
| sentiment | categorical/3 | 1660 | 33 | 6/6 | 291/175 | *irish economic sentiment analysis* [1] Articles from three Irish online news sources were annotated by volunteer users as positive, negative, or irrelevant. |
| wosi | ordinal/[0 10] | 30 | 10 | 10/10 | 30/30 | *word similarity* [10] Numeric judgements of word similarity. |
| valence | ordinal[-100 100] | 100 | 38 | 10/10 | 26/20 | *affect recognition* [10] Each annotator is presented with a short headline and asked to rate it on a scale [-100,100] to denote the overall positive or negative valence. |

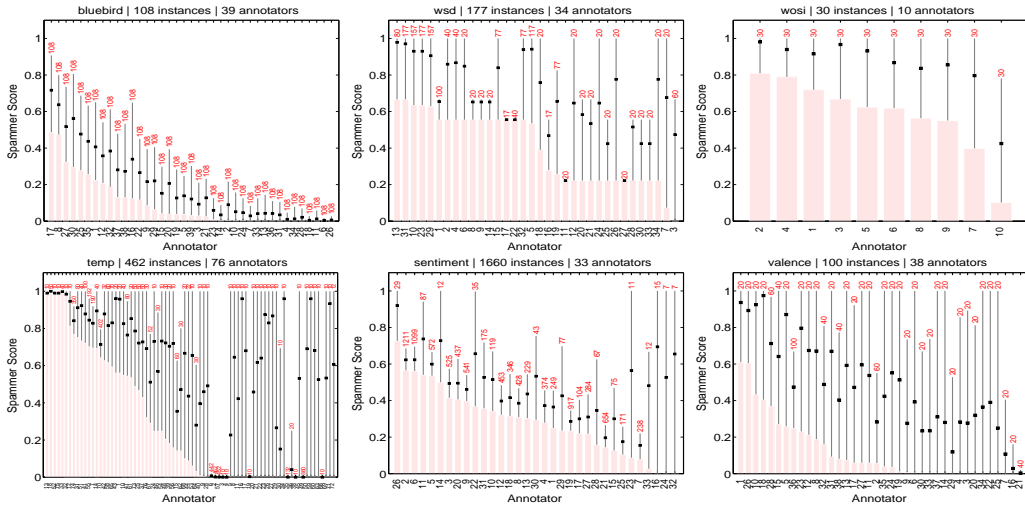

Figure 4: *Annotator Rankings* The rankings obtained for the datasets in Table 1. The spammer score ranges from 0 to 1, the lower the score, the more spammy the annotator. The mean spammer score and the 95% confidence intervals (CI) are shown—obtained from 100 bootstrap replications. The annotators are ranked based on the lower limit of the 95% CI. The number at the top of the CI bar shows the number of instances annotated by that annotator. Note that the CIs are wider when the annotator labels only a few instances.

when the annotator labels only a few instances. For a crowdsourced labeling task the annotator has to be good and also label a reasonable number of instances in order to be reliably identified.

**Simulated data** We first illustrate our proposed spammer score on simulated binary data (with equal prevalence for both classes) consisting of 500 instances labeled by 30 annotators of varying sensitivity and specificity (see Figure 3(a) for the simulation setup). Of the 30 annotators we have 10 good annotators (annotators 1 to 10 who lie above the diagonal in Figure 3(a)), 10 spammers (annotators 11 to 20 who lie around the diagonal), and 10 malicious annotators (annotators 21 to 30 who lie below the diagonal). Figure 3(b) plots the ranking of annotators obtained using the proposed spammer score with the annotator model parameters estimated via the EM algorithm [3, 7]. The spammer score ranges from 0 to 1, the lower the score, the more spammy the annotator. The mean spammer score and the 95% confidence interval (CI) obtained via bootstrapping are shown. The annotators are ranked based on the lower limit of the 95% CI. As can be seen all the spammers (annotators 11 to 20) have a low spammer score and appear at the bottom of the list. The malicious annotators have higher score than the spammers since we can correct for their flipping. The malicious annotators are good annotators but they flip their labels and as such are not spammers if we detect that they are malicious. Figure 3(c) compares the (median) rank obtained via the spammer score with the (median) rank obtained using accuracy as the score to rank the annotators. While the good annotators are ranked high by both methods the accuracy score gives a low rank to the malicious annotators. Accuracy does not capture the notion of a spammer. Figure 3(d) compares the ranking with the method proposed by Ipeirotis et. al. [4] which gives almost similar rankings as our proposed score.

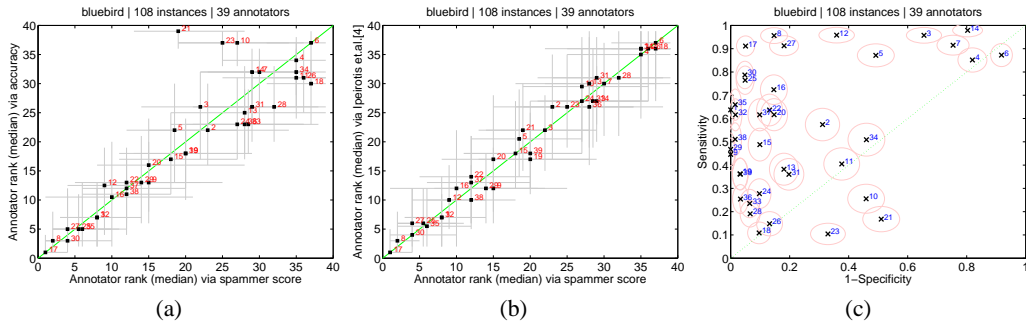

Figure 5: Comparison of the rank obtained via the spammer score with the rank obtained using (a) accuracy and (b) the method proposed by Ipeirotis et. al. [4] for the bluebird binary dataset. (c) The annotator model parameters as estimated by the EM algorithm [3, 7].

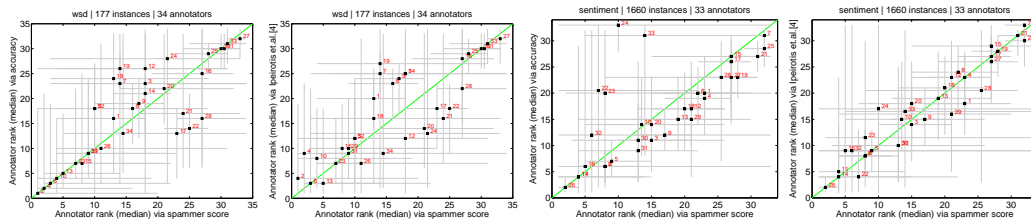

Figure 6: Comparison of the median rank obtained via the spammer score with the rank obtained using accuracy and he method proposed by Ipeirotis et. al. [4] for the two categorial datasets in Table 1.

**Mechanical Turk data** We report results on some publicly available linguistic and image annotation data collected using the Amazon's Mechanical Turk (AMT) and other sources. Table 1 summarizes the datasets. Figure 4 plots the spammer scores and rankings obtained. The mean and the $95\%$ CI obtained via bootstrapping are also shown. The number at the top of the CI bar shows the number of instances annotated by that annotator. The rankings are based on the lower limit of the $95\%$ CI which factors the number of instances labeled by the annotator into the ranking. An annotator who labels only a few instances will have very wide CI. Some annotators who label only a few instances may have a high mean spammer score but the CI will be wide and hence ranked lower. Ideally we would like to have annotators with a high score and at the same time label a lot of instances so that we can reliablly identify them. The authors [1] for the sentiment dataset shared with us some of the qualitative observations regarding the annotators and they somewhat agree with our rankings. For example the authors made the following comments about Annotator 7 *"Quirky annotator - had a lot of debate about what was the meaning of the annotation question. I'd say he changed his labeling strategy at least once during the process"*. Our proposed score gave a low rank to this annotator.

**Comparison with other approaches** Figure 5 and 6 compares the proposed ranking with the rank obtained using accuracy and the method proposed by Ipeirotis et. al. [4] for some binary and categorical datasets in Table 1. Our proposed ranking is somewhat similar to that obtained by Ipeirotis et. al. [4] but accuracy does not quite capture the notion of spammer. For example for the bluebird dataset for annotator 21 (see Figure 5(a)) accuracy ranks it at the bottom of the list while the proposed score puts is in the middle of the list. From the estimated model parameters it can be seen that annotator 21 actually flips the labels (below the diagonal in Figure 5(c)) but is a good annotator.

# 7 Conclusions

We proposed a score to rank annotators for crowdsourced binary, categorical, and ordinal labeling tasks. The obtained rankings and the scores can be used to allocate monetary bonuses to be paid to different annotators and also to eliminate spammers from further labeling tasks. A mechanism to rank annotators should be desirable feature of any crowdsourcing service. The proposed score should also be useful to specify the prior for Bayesian approaches to consolidate annotations.

## Footnotes

[1]One of the commonly used strategy to filter out spammers is to inject some items into the annotations with known labels. This is the strategy used by CrowdFlower (http://crowdflower.com/docs/gold).

[2]Also note that $(\alpha^j + \beta^j)/2$ is equal to the area shown in the plot and can be considered as a non-parametric approximation to the area under the ROC curve (AUC) based on one observed point. It is also equal to the Balanced Classification Rate (BCR). So a spammer can also be defined as having BCR or AUC equal to 0.5.

[3] This can be seen as follows: $\Pr[y_i^j = k \mid y_i = 1] = \Pr[(y_i^j \geq k) \text{ AND } (y_i^j < k+1) \mid y_i = 1] = \Pr[y_i^j \geq k \mid y_i = 1] + \Pr[y_i^j < k+1 \mid y_i = 1] - \Pr[(y_i^j \geq k) \text{ OR } (y_i^j < k+1) \mid y_i = 1] = \Pr[y_i^j \geq k \mid y_i = 1] - \Pr[y_i^j \geq k+1 \mid y_i = 1] = \alpha_k^j - \alpha_{k+1}^j$. Here we used the fact that $\Pr[(y_i^j \geq k) \text{ OR } (y_i^j < k+1)] = 1$.

# References

[1] A. Brew, D. Greene, and P. Cunningham. Using crowdsourcing and active learning to track sentiment in online media. In *Proceedings of the 6th Conference on Prestigious Applications of Intelligent Systems (PAIS'10)*, 2010.

[2] B. Carpenter. Multilevel bayesian models of categorical data annotation. Technical Report available at http://lingpipe-blog.com/lingpipe-white-papers/, 2008.

[3] A. P. Dawid and A. M. Skene. Maximum likeihood estimation of observer error-rates using the EM algorithm. *Applied Statistics*, 28(1):20–28, 1979.

[4] P. G. Ipeirotis, F. Provost, and J. Wang. Quality management on Amazon Mechanical Turk. In *Proceedings of the ACM SIGKDD Workshop on Human Computation (HCOMP'10)*, pages 64–67, 2010.

[5] V. C. Raykar and S. Yu. An entropic score to rank annotators for crowdsourced labelling tasks. In *Proceedings of the Third National Conference on Computer Vision, Pattern Recognition, Image Processing and Graphics (NCVPRIPG)*, 2011.

[6] V. C. Raykar, S. Yu, L .H. Zhao, A. Jerebko, C. Florin, G. H. Valadez, L. Bogoni, and L. Moy. Supervised learning from multiple experts: Whom to trust when everyone lies a bit. In *Proceedings of the 26th International Conference on Machine Learning (ICML 2009)*, pages 889–896, 2009.

[7] V. C. Raykar, S. Yu, L. H. Zhao, G. H. Valadez, C. Florin, L. Bogoni, and L. Moy. Learning from crowds. *Journal of Machine Learning Research*, 11:1297–1322, April 2010.

[8] V. S. Sheng, F. Provost, and P. G. Ipeirotis. Get another label? Improving data quality and data mining using multiple, noisy labelers. In *Proceedings of the 14th ACM SIGKDD International Conference on Knowledge Discovery and Data Mining*, pages 614–622, 2008.

[9] P. Smyth, U. Fayyad, M. Burl, P. Perona, and P. Baldi. Inferring ground truth from subjective labelling of venus images. In *Advances in Neural Information Processing Systems 7*, pages 1085–1092. 1995.

[10] R. Snow, B. O'Connor, D. Jurafsky, and A. Y. Ng. Cheap and Fast—but is it good? Evaluating Non-Expert Annotations for Natural Language Tasks. In *Proceedings of the Conference on Empirical Methods in Natural Language Processing (EMNLP '08)*, pages 254–263, 2008.

[11] A. Sorokin and D. Forsyth. Utility data annotation with Amazon Mechanical Turk. In *Proceedings of the First IEEE Workshop on Internet Vision at CVPR 08*, pages 1–8, 2008.

[12] P. Welinder, S. Branson, S. Belongie, and P. Perona. The multidimensional wisdom of crowds. In *Advances in Neural Information Processing Systems 23*, pages 2424–2432. 2010.

[13] J. Whitehill, P. Ruvolo, T. Wu, J. Bergsma, and J. Movellan. Whose vote should count more: Optimal integration of labels from labelers of unknown expertise. In *Advances in Neural Information Processing Systems 22*, pages 2035–2043. 2009.

[14] Y. Yan, R. Rosales, G. Fung, M. Schmidt, G. Hermosillo, L. Bogoni, L. Moy, and J. Dy. Modeling annotator expertise: Learning when everybody knows a bit of something. In *Proceedings of the Thirteenth International Conference on Artificial Intelligence and Statistics (AISTATS 2010)*, pages 932–939, 2010.

